# Collapsed Variational Inference for HDP

**Yee Whye Teh**
Gatsby Unit
University College London
ywteh@gatsby.ucl.ac.uk

**Kenichi Kurihara**
Dept. of Computer Science
Tokyo Institute of Technology
kurihara@mi.cs.titech.ac.jp

**Max Welling**
ICS
UC Irvine
welling@ics.uci.edu

## Abstract

A wide variety of Dirichlet-multinomial 'topic' models have found interesting applications in recent years. While Gibbs sampling remains an important method of inference in such models, variational techniques have certain advantages such as easy assessment of convergence, easy optimization without the need to maintain detailed balance, a bound on the marginal likelihood, and side-stepping of issues with topic-identifiability. The most accurate variational technique thus far, namely collapsed variational latent Dirichlet allocation, did not deal with model selection nor did it include inference for hyperparameters. We address both issues by generalizing the technique, obtaining the first variational algorithm to deal with the hierarchical Dirichlet process and to deal with hyperparameters of Dirichlet variables. Experiments show a significant improvement in accuracy.

## 1 Introduction

Many applications of graphical models have traditionally dealt with discrete state spaces, where each variable is multinomial distributed given its parents [1]. Without strong prior knowledge on the structure of dependencies between variables and their parents, the typical Bayesian prior over parameters has been the Dirichlet distribution. This is because the Dirichlet prior is conjugate to the multinomial, leading to simple and efficient computations for both the posterior over parameters and the marginal likelihood of data. When there are latent or unobserved variables, the variational Bayesian approach to posterior estimation, where the latent variables are assumed independent from the parameters, has proven successful [2].

In recent years there has been a proliferation of graphical models composed of a multitude of multinomial and Dirichlet variables interacting in various inventive ways. The major classes include the latent Dirichlet allocation (LDA) [3] and many other topic models inspired by LDA, and the hierarchical Dirichlet process (HDP) [4] and many other nonparametric models based on the Dirichlet process (DP). LDA pioneered the use of Dirichlet distributed latent variables to represent shades of membership to different clusters or topics, while the HDP pioneered the use of nonparametric models to sidestep the need for model selection.

For these Dirichlet-multinomial models the inference method of choice is typically collapsed Gibbs sampling, due to its simplicity, speed, and good predictive performance on test sets. However there are drawbacks as well: it is often hard to access convergence of the Markov chains, it is harder still to accurately estimate the marginal probability of the training data or the predictive probability of test data (if latent variables are associated with the test data), averaging topic-dependent quantities based on samples is not well-defined because the topic labels may have switched during sampling and avoiding local optima through large MCMC moves such as split and merge algorithms are tricky to implement due to the need to preserve detailed balance. Thus there seems to be a genuine need to consider alternatives to sampling.

For LDA and its cousins, there are alternatives based on variational Bayesian (VB) approximations [3] and on expectation propagation (EP) [5]. [6] found that EP was not efficient enough for large

scale applications, while VB suffered from significant bias resulting in worse predictive performance than Gibbs sampling. [7] addressed these issues by proposing an improved VB approximation based on the idea of collapsing, that is, integrating out the parameters while assuming that other latent variables are independent. As for nonparametric models, a number of VB approximations have been proposed for DP mixture models [8, 9], while to our knowledge none has been proposed for the HDP thus far ([10] derived a VB inference for the HDP, but dealt only with point estimates for higher level parameters).

In this paper we investigate a new VB approach to inference for the class of Dirichlet-multinomial models. To be concrete we focus our attention on an application of the HDP to topic modeling [4], though the approach is more generally applicable. Our approach is an extension of the collapsed VB approximation for LDA (CV-LDA) presented in [7], and represents the first VB approximation to the HDP[1]. We call this the collapsed variational HDP (CV-HDP). The advantage of CV-HDP over CV-LDA is that the optimal number of variational components is not finite. This implies, apart from local optima, that we can keep adding components indefinitely while the algorithm will take care removing unnecessary clusters. Ours is also the first variational algorithm to treat full posterior distributions over the hyperparameters of Dirichlet variables, and we show experimentally that this results in significant improvements in both the variational bound and test-set likelihood. We expect our approach to be generally applicable to a wide variety of Dirichlet-multinomial models beyond what we have described here.

## 2 A Nonparametric Hierarchical Bayesian Topic Model

We consider a document model where each document in a corpus is modelled as a mixture over topics, and each topic is a distribution over words in the vocabulary. Let there be $D$ documents in the corpus, and $W$ words in the vocabulary. For each document $d = 1, \ldots, D$, let $\theta_d$ be a vector of mixing proportions over topics. For each topic $k$, let $\phi_k$ be a vector of probabilities for words in that topic. Words in each document are drawn as follows: first choose a topic $k$ with probability $\theta_{dk}$, then choose a word $w$ with probability $\phi_{kw}$. Let $x_{id}$ be the $i$th word token in document $d$, and $z_{id}$ its chosen topic. We have,

$$z_{id} \,|\, \theta_d \sim \text{Mult}(\theta_d) \qquad\qquad x_{id} \,|\, z_{id}, \phi_{z_{id}} \sim \text{Mult}(\phi_{z_{id}}) \qquad\qquad (1)$$

We place Dirichlet priors on the parameters $\theta_d$ and $\phi_k$,

$$\theta_d \,|\, \pi \sim \text{Dir}(\alpha\pi) \qquad\qquad \phi_k \,|\, \tau \sim \text{Dir}(\beta\tau) \qquad\qquad (2)$$

where $\pi$ is the corpus-wide distribution over topics, $\tau$ is the corpus-wide distribution over the vocabulary, and $\alpha$ and $\beta$ are concentration parameters describing how close $\theta_d$ and $\phi_k$ are to their respective prior means $\pi$ and $\tau$.

If the number of topics $K$ is finite and fixed, the above model is LDA. As we usually do not know the number of topics a priori, and would like a model that can determine this automatically, we consider a nonparametric extension reposed on the HDP [4]. Specifically, we have a countably infinite number of topics (thus $\theta_d$ and $\pi$ are infinite-dimensional vectors), and we use a stick-breaking representation [11] for $\pi$:

$$\pi_k = \tilde{\pi}_k \prod_{l=1}^{k-1}(1 - \tilde{\pi}_l) \qquad\qquad \tilde{\pi}_k|\gamma \sim \text{Beta}(1, \gamma) \qquad\qquad \text{for } k = 1, 2, \ldots \qquad (3)$$

In the normal Dirichlet process notation, we would equivalently have $G_d \sim \text{DP}(\alpha, G_0)$ and $G_0 \sim \text{DP}(\gamma, \text{Dir}(\beta\tau))$, where $G_d = \sum_{k=1}^{\infty} \theta_{dk}\delta_{\phi_k}$ and $G_0 = \sum_{k=1}^{\infty} \pi_k\delta_{\phi_k}$ are sums of point masses, and $\text{Dir}(\beta\tau)$ is the base distribution. Finally, in addition to the prior over $\pi$, we place priors over the other hyperparameters $\alpha, \beta, \gamma$ and $\tau$ of the model as well,

$$\alpha \sim \text{Gamma}(a_\alpha, b_\alpha) \qquad \beta \sim \text{Gamma}(a_\beta, b_\beta) \qquad \gamma \sim \text{Gamma}(a_\gamma, b_\gamma) \qquad \tau \sim \text{Dir}(a_\tau) \qquad (4)$$

The full model is shown graphically in Figure 1(left).

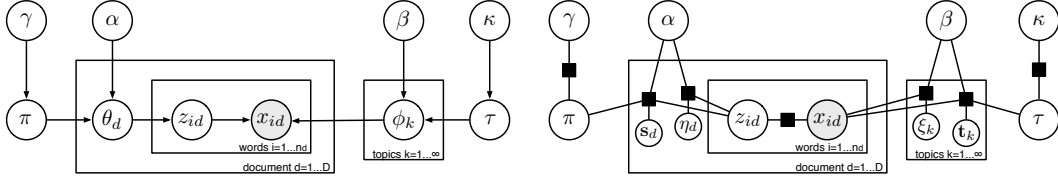

Figure 1: Left: The HDP topic model. Right: Factor graph of the model with auxiliary variables.

## 3 Collapsed Variational Bayesian Inference for HDP

There is substantial empirical evidence that marginalizing out variables is helpful for efficient inference. For instance, in [12] it was observed that Gibbs sampling enjoys better mixing, while in [7] it was shown that variational inference is more accurate in this collapsed space. In the following we will build on this experience and propose a collapsed variational inference algorithm for the HDP, based upon first replacing the parameters with auxiliary variables, then effectively collapsing out the auxiliary variables variationally. The algorithm is fully Bayesian in the sense that all parameter posteriors are treated exactly and full posterior distributions are maintained for all hyperparameters. The only assumptions made are independencies among the latent topic variables and hyperparameters, and that there is a finite upper bound on the number of topics used (which is found automatically). The only inputs required of the modeller are the values of the top-level parameters $a_\alpha, b_\alpha, ...$.

### 3.1 Replacing parameters with auxiliary variables

In order to obtain efficient variational updates, we shall replace the parameters $\boldsymbol{\theta} = \{\theta_d\}$ and $\boldsymbol{\phi} = \{\phi_k\}$ with auxiliary variables. Specifically, we first integrate out the parameters; this gives a joint distribution over latent variables $\mathbf{z} = \{z_{id}\}$ and word tokens $\mathbf{x} = \{x_{id}\}$ as follows:

$$p(\mathbf{z}, \mathbf{x}|\alpha, \beta, \gamma, \pi, \tau) = \prod_{d=1}^{D} \frac{\Gamma(\alpha)}{\Gamma(\alpha+n_{d\cdot\cdot})} \prod_{k=1}^{K} \frac{\Gamma(\alpha\pi_k+n_{dk\cdot})}{\Gamma(\alpha\pi_k)} \prod_{k=1}^{K} \frac{\Gamma(\beta)}{\Gamma(\beta+n_{\cdot k\cdot})} \prod_{w=1}^{W} \frac{\Gamma(\beta\tau_w+n_{\cdot kw})}{\Gamma(\beta\tau_w)} \quad (5)$$

with $n_{dkw} = \#\{i : x_{id} = w, z_{id} = k\}$, dot denoting sum over that index, and $K$ denoting an index such that $z_{id} \leq K$ for all $i, d$. The ratios of gamma functions in (5) result from the normalization constants of the Dirichlet densities of $\boldsymbol{\theta}$ and $\boldsymbol{\phi}$, and prove to be nuisances for updating the hyperparameter posteriors. Thus we introduce four sets of auxiliary variables: $\eta_d$ and $\xi_k$ taking values in $[0, 1]$, and $s_{dk}$ and $t_{kw}$ taking integral values. This results in a joint probability distribution over an expanded system,

$$p(\mathbf{z}, \mathbf{x}, \boldsymbol{\eta}, \boldsymbol{\xi}, \mathbf{s}, \mathbf{t}|\alpha, \beta, \gamma, \pi, \tau)$$
$$= \prod_{d=1}^{D} \frac{\eta_d^{\alpha-1}(1-\eta_d)^{n_{d\cdot\cdot}-1} \prod_{k=1}^{K} {n_{dk\cdot} \brack s_{dk}} (\alpha\pi_k)^{s_{dk}}}{\Gamma(n_{d\cdot\cdot})} \prod_{k=1}^{K} \frac{\xi_k^{\beta-1}(1-\xi_k)^{n_{\cdot k\cdot}-1} \prod_{w=1}^{W} {n_{\cdot kw} \brack t_{kw}} (\beta\tau_w)^{t_{kw}}}{\Gamma(n_{\cdot k\cdot})} \quad (6)$$

where ${n \brack m}$ are unsigned Stirling numbers of the first kind, and bold face letters denote sets of the corresponding variables. It can be readily verified that marginalizing out $\boldsymbol{\eta}$, $\boldsymbol{\xi}$, $\mathbf{s}$ and $\mathbf{t}$ reduces (6) to (5). The main insight is that conditioned on $\mathbf{z}$ and $\mathbf{x}$ the auxiliary variables are independent and have well-known distributions. Specifically, $\eta_d$ and $\xi_k$ are Beta distributed, while $s_{dk}$ (respectively $t_{kw}$) is the random number of occupied tables in a Chinese restaurant process with $n_{dk\cdot}$ (respectively $n_{\cdot kw}$) customers and a strength parameter of $\alpha\pi_k$ (respectively $\beta\tau_w$) [13, 4].

### 3.2 The Variational Approximation

We assume the following form for the variational posterior over the auxiliary variables system:

$$q(\mathbf{z}, \boldsymbol{\eta}, \boldsymbol{\xi}, \mathbf{s}, \mathbf{t}, \alpha, \beta, \gamma, \tau, \pi) = q(\alpha)q(\beta)q(\gamma)q(\tau)q(\pi)q(\boldsymbol{\eta}, \boldsymbol{\xi}, \mathbf{s}, \mathbf{t}|\mathbf{z}) \prod_{d=1}^{D} \prod_{i=1}^{n_{d\cdot\cdot}} q(z_{id}) \quad (7)$$

where the dependence of auxiliary variables on $\mathbf{z}$ is modelled exactly. [7] showed that modelling exactly the dependence of a set of variables on another set is equivalent to integrating out the first

set. Thus we can interpret (7) as integrating out the auxiliary variables with respect to $\mathbf{z}$. Given the above factorization, $q(\pi)$ further factorizes so that the $\tilde{\pi}_k$'s are independent, as do the posterior over auxiliary variables.

For computational tractability, we also truncated our posterior representation to $K$ topics. Specifically, we assumed that $q(z_{id} > K) = 0$ for every $i$ and $d$. A consequence is that observations have no effect on $\tilde{\pi}_k$ and $\phi_k$ for all $k > K$, and these parameters can be exactly marginalized out. Notice that our approach to truncation is different from that in [8], who implemented a truncation at $T$ by instead fixing the posterior for the stick weight $q(v_T = 1) = 1$, and from that in [9], who assumed that the variational posteriors for parameters beyond the truncation level are set at their priors. Our truncation approximation is nested like that in [9], and unlike that in [8]. Our approach is also simpler than that in [9], which requires computing an infinite sum which is intractable in the case of HDPs. We shall treat $K$ as a parameter of the variational approximation, possibly optimized by iteratively splitting or merging topics (though we have not explored these in this paper; see discussion section). As in [9], we reordered the topic labels such that $\mathbb{E}[n_{\cdot 1 \cdot}] > \mathbb{E}[n_{\cdot 2 \cdot}] > \cdots$. An expression for the variational bound on the marginal log-likelihood is given in appendix A.

### 3.3 Variational Updates

In this section we shall derive the complete set of variational updates for the system. In the following $\mathbb{E}[y]$ denotes the expectation of $y$, $\mathbb{G}[y] = e^{\mathbb{E}[\log y]}$ the geometric expectation, and $\mathbb{V}[y] = \mathbb{E}[y^2] - \mathbb{E}[y]^2$ the variance. Let $\Psi(y) = \frac{\partial \log \Gamma(y)}{\partial y}$ be the digamma function. We shall also employ index summation shorthands: $\cdot$ sums out that index, while $_{>l}$ sums over $i$ where $i > l$.

*Hyperparameters.* Updates for the hyperparameters are derived using the standard fully factorized variational approach, since they are assumed independent from each other and from other variables. For completeness we list these here, noting that $\alpha$, $\beta$, $\gamma$ are gamma distributed in the posterior, $\tilde{\pi}_k$'s are beta distributed, and $\tau$ is Dirichlet distributed:

$$q(\alpha) \propto \alpha^{a_\alpha + \mathbb{E}[s_{\cdot\cdot}] - 1} e^{-\alpha(b_\alpha - \sum_d \mathbb{E}[\log \eta_d])} \qquad q(\tilde{\pi}_k) \propto \tilde{\pi}_k^{\mathbb{E}[s_{\cdot k}]}(1 - \tilde{\pi}_k)^{\mathbb{E}[\gamma] + \mathbb{E}[s_{\cdot > k}] - 1} \qquad (8)$$

$$q(\beta) \propto \beta^{a_\beta + \mathbb{E}[t_{\cdot\cdot}] - 1} e^{-\beta(b_\beta - \sum_k \mathbb{E}[\log \xi_k])} \qquad q(\tau) \propto \prod_{w=1}^W \tau_w^{a_\tau + \mathbb{E}[t_{\cdot w}] - 1}$$

$$q(\gamma) \propto \gamma^{a_\gamma + K - 1} e^{-\gamma(b_\gamma - \sum_{k=1}^K \mathbb{E}[\log(1 - \tilde{\pi}_k)])}$$

In subsequent updates we will need averages and geometric averages of these quantities which can be extracted using the following identities: $p(x) \propto x^{a-1} e^{-bx} \Rightarrow \mathbb{E}[x] = a/b$, $\mathbb{G}[x] = e^{\Psi(a)}/b$, $p(x) \propto \prod_k x_k^{a_k - 1} \Rightarrow \mathbb{G}[x_k] = e^{\Psi(a_k)}/e^{\Psi(\sum_k a_k)}$. Note also that the geometric expectations factorizes: $\mathbb{G}[\alpha\pi_k] = \mathbb{G}[\alpha]\mathbb{G}[\pi_k]$, $\mathbb{G}[\beta\tau_w] = \mathbb{G}[\beta]\mathbb{G}[\tau_w]$ and $\mathbb{G}[\pi_k] = \mathbb{G}[\tilde{\pi}_k] \prod_{l=1}^{k-1} \mathbb{G}[1 - \tilde{\pi}_l]$.

*Auxiliary variables.* The variational posteriors for the auxiliary variables depend on $\mathbf{z}$ through the counts $n_{dkw}$. $\eta_d$ and $\xi_k$ are beta distributed. If $n_{dk\cdot} = 0$ then $q(s_{dk} = 0) = 1$ otherwise $q(s_{dk}) > 0$ only if $1 \le s_{dk} \le n_{dk\cdot}$. Similarly for $t_{kw}$. The posteriors are:

$$q(\eta_d | \mathbf{z}) \propto \eta_d^{\mathbb{E}[\alpha] - 1}(1 - \eta_d)^{n_{d\cdot\cdot} - 1} \qquad q(s_{dk} = m | \mathbf{z}) \propto \begin{bmatrix} n_{dk\cdot} \\ m \end{bmatrix} (\mathbb{G}[\alpha\pi_k])^m \qquad (9)$$

$$q(\xi_k | \mathbf{z}) \propto \xi_k^{\mathbb{E}[\beta] - 1}(1 - \xi_k)^{n_{\cdot k\cdot} - 1} \qquad q(t_{kw} = m | \mathbf{z}) \propto \begin{bmatrix} n_{\cdot kw} \\ m \end{bmatrix} (\mathbb{G}[\beta\tau_w])^m$$

To obtain expectations of the auxiliary variables in (8) we will have to average over $\mathbf{z}$ as well. For $\eta_d$ this is $\mathbb{E}[\log \eta_d] = \Psi(\mathbb{E}[\alpha]) - \Psi(\mathbb{E}[\alpha] + n_{d\cdot\cdot})$ where $n_{d\cdot\cdot}$ is the (fixed) number of words in document $d$. For the other auxiliary variables these expectations depend on counts which can take on many values and a naïve computation can be expensive. We derive computationally tractable approximations based upon an improvement to the second-order approximation in [7]. As we see in the experiments these approximations are very accurate. Consider $\mathbb{E}[\log \xi_k]$. We have,

$$\mathbb{E}[\log \xi_k | \mathbf{z}] = \Psi(\mathbb{E}[\beta]) - \Psi(\mathbb{E}[\beta] + n_{\cdot k\cdot}) \qquad (10)$$

and we need to average over $n_{\cdot k\cdot}$ as well. [7] tackled a similar problem with $\log$ instead of $\Psi$ using a second order Taylor expansion to $\log$. Unfortunately such an approximation failed to work in our case as the digamma function $\Psi(y)$ diverges much more quickly than $\log y$ at $y = 0$. Our solution is to treat the case $n_{\cdot k\cdot} = 0$ exactly, and apply the second-order approximation when $n_{\cdot k\cdot} > 0$. This leads to the following approximation:

$$\mathbb{E}[\log \xi_k] \approx \mathbb{P}_+[n_{\cdot k\cdot}] \left( \Psi(\mathbb{E}[\beta]) - \Psi(\mathbb{E}[\beta] + \mathbb{E}_+[n_{\cdot k\cdot}]) - \tfrac{1}{2}\mathbb{V}_+[n_{\cdot k\cdot}]\Psi''(\mathbb{E}[\beta] + \mathbb{E}_+[n_{\cdot k\cdot}]) \right) \qquad (11)$$

where $\mathbb{P}_+$ is the "probability of being positive" operator: $\mathbb{P}_+[y] = q(y > 0)$, and $\mathbb{E}_+[y], \mathbb{V}_+[y]$ are the expectation and variance conditional on $y > 0$. The other two expectations are derived similarly, making use of the fact that $s_{dk}$ and $t_{kw}$ are distributionally equal to the random numbers of tables in Chinese restaurant processes:

$$\mathbb{E}[s_{dk}] \approx \mathbb{G}[\alpha\pi_k]\mathbb{P}_+[n_{dk\cdot}]\left(\Psi(\mathbb{G}[\alpha\pi_k]+\mathbb{E}_+[n_{dk\cdot}])-\Psi(\mathbb{G}[\alpha\pi_k])+\tfrac{\mathbb{V}_+[n_{dk\cdot}]\Psi''(\mathbb{G}[\alpha\pi_k]+\mathbb{E}_+[n_{dk\cdot}])}{2}\right) \quad (12)$$

$$\mathbb{E}[t_{kw}] \approx \mathbb{G}[\beta\tau_w]\mathbb{P}_+[n_{\cdot kw}]\left(\Psi(\mathbb{G}[\beta\tau_w]+\mathbb{E}_+[n_{\cdot kw}])-\Psi(\mathbb{G}[\beta\tau_w])+\tfrac{\mathbb{V}_+[n_{\cdot kw}]\Psi''(\mathbb{G}[\beta\tau_w]+\mathbb{E}_+[n_{\cdot kw}])}{2}\right)$$

As in [7], we can efficiently track the relevant quantities above by noting that each count is a sum of independent Bernoulli variables. Consider $n_{dk\cdot}$ as an example. We keep track of three quantities:

$$\mathbb{E}[n_{dk\cdot}] = \sum_i q(z_{id}=k) \quad \mathbb{V}[n_{dk\cdot}] = \sum_i q(z_{id}=k)q(z_{id}\neq k) \quad \mathbb{Z}[n_{dk\cdot}] = \sum_i \log q(z_{id}\neq k) \quad (13)$$

Some algebraic manipulations now show that:

$$\mathbb{P}_+[n_{dk\cdot}] = 1 - e^{\mathbb{Z}[n_{dk\cdot}]} \quad \mathbb{E}_+[n_{dk\cdot}] = \frac{\mathbb{E}[n_{dk\cdot}]}{\mathbb{P}_+[n_{dk\cdot}]} \quad \mathbb{V}_+[n_{dk\cdot}] = \frac{\mathbb{V}[n_{dk\cdot}]}{\mathbb{P}_+[n_{dk\cdot}]} - e^{\mathbb{Z}[n_{dk\cdot}]}\mathbb{E}_+[n_{dk\cdot}] \quad (14)$$

*Topic assignment variables.* [7] showed that if the dependence of a set of variables, say $\mathbf{A}$, on another set of variables, say $\mathbf{z}$, is modelled exactly, then in deriving the updates for $\mathbf{z}$ we may equivalently integrate out $\mathbf{A}$. Applying to our situation with $\mathbf{A} = \{\boldsymbol{\eta}, \boldsymbol{\xi}, \mathbf{s}, \mathbf{t}\}$, we obtain updates similar to those in [7], except that the hyperparameters are replaced by either their expectations or their geometric expectations, depending on which is used in the updates for the corresponding auxiliary variables:

$$q(z_{id}=k) \propto \mathbb{G}\big[\mathbb{G}[\alpha\pi_k]+n_{dk\cdot}^{\neg id}\big]\mathbb{G}\big[\mathbb{G}[\beta\tau_{x_{id}}]+n_{\cdot kx_{id}}^{\neg id}\big]\mathbb{G}\big[\mathbb{E}[\beta]+n_{\cdot k\cdot}^{\neg id}\big]^{-1}$$

$$\approx \propto \big(\mathbb{G}[\alpha\pi_k]+\mathbb{E}[n_{dk\cdot}^{\neg id}]\big)\big(\mathbb{G}[\beta\tau_{x_{id}}]+\mathbb{E}[n_{\cdot kx_{id}}^{\neg id}]\big)\big(\mathbb{E}[\beta]+\mathbb{E}[n_{\cdot k\cdot}^{\neg id}]\big)^{-1}$$

$$\exp\left(-\frac{\mathbb{V}[n_{dk\cdot}^{\neg id}]}{2(\mathbb{G}[\alpha\pi_k]+\mathbb{E}[n_{dk\cdot}^{\neg id}])^2} - \frac{\mathbb{V}[n_{\cdot kx_{id}}^{\neg id}]}{2(\mathbb{G}[\beta\tau_{x_{id}}]+\mathbb{E}[n_{\cdot kx_{id}}^{\neg id}])^2} + \frac{\mathbb{V}[n_{\cdot k\cdot}^{\neg id}]}{2(\mathbb{E}[\beta]+\mathbb{E}[n_{\cdot k\cdot}^{\neg id}])^2}\right) \quad (15)$$

## 4 Experiments

We implemented and compared performances for 5 inference algorithms for LDA and HDP: 1) variational LDA (V-LDA) [3], collapsed variational LDA (CV-LDA) [7], collapsed variational HDP (CV-HDP, this paper), collapsed Gibbs sampling for LDA (G-LDA) [12] and the direct assignment Gibbs sampler for HDP (G-HDP) [4].

We report results on the following 3 datasets: i) KOS ($W = 6906$, $D = 3430$, number of word-tokens $N = 467,714$), ii) a subset of the Reuters dataset consisting of news-topics with a number of documents larger than 300 ($W = 4593$, $D = 8433$, $N = 566,298$), iii) a subset of the 20News-groups dataset consisting of the topics 'comp.os.ms-windows.misc', 'rec.autos', 'rec.sport.baseball', 'sci.space' and 'talk.politics.misc' ($W = 8424$, $D = 4716$, $N = 437,850$).

For G-HDP we use the released code at http://www.gatsby.ucl.ac.uk/~ywteh/research/software.html. The variables $\beta, \tau$ are not adapted in that code, so we fixed them at $\beta = 100$ and $\tau_w = 1/W$ for all algorithms (see below for discussion regarding adapting these in CV-HDP). G-HDP was initialized with either 1 topic (G-HDP$_1$) or with 100 topics (G-HDP$_{100}$). For CV-HDP we use the following initialization: $\mathbb{E}[\beta] = \mathbb{G}[\beta] = 100$ and $\mathbb{G}[\tau_w] = 1/W$ (kept fixed to compare with G-HDP), $\mathbb{E}[\alpha] = a_\alpha/b_\alpha$, $\mathbb{G}[\alpha] = e^{\Psi(a_\alpha)}/b_\alpha$, $\mathbb{E}[\gamma] = a_\gamma/b_\gamma$, $\mathbb{G}[\pi_k] = 1/K$ and $q(z_{ij}=k) \propto 1 + u$ with $u \sim \mathcal{U}[0,1]$. We set[2] hyperparameters $a_\alpha, b_\alpha, a_\beta, b_\beta$ in the range between [2, 6], while $a_\gamma, b_\gamma$ was chosen in the range [5, 10] and $a_\tau$ in $[30-50]/W$. The number of topics used in CV-HDP was truncated at 40, 80, and 120 topics, corresponding to the number of topics used in the LDA algorithms. Finally, for all LDA algorithms we used $\alpha = 0.1$, $\pi = 1/K$.

Performance was evaluated by comparing i) the in-sample (train) variational bound on the log-likelihood for all three variational methods and ii) the out-of-sample (test) log-likelihood for all five methods. All inference algorithms were run on 90% of the words in each document while test-set performance was evaluated on the remaining 10% of the words. Test-set log-likelihood was computed as follows for the variational methods:

$$p(\mathbf{x}^{\text{test}}) = \prod_{ij} \sum_k \bar{\theta}_{jk} \bar{\phi}_{kx_{ij}^{\text{test}}} \qquad \bar{\theta}_{jk} = \frac{\alpha\pi_k + E_q[n_{jk\cdot}]}{\alpha + E_q[n_{j\cdot\cdot}]} \qquad \bar{\phi}_{kw} = \frac{\beta\tau_w + E_q[n_{\cdot kw}]}{\beta + E_q[n_{\cdot k\cdot}]} \qquad (16)$$

Note that we used estimated mean values of $\theta_{jk}$ and $\phi_{kw}$ [14]. For CV-HDP we replaced all hyperparameters by their expectations. For the Gibbs sampling algorithms, given $S$ samples from the posterior, we used:

$$p(\mathbf{x}^{\text{test}}) = \prod_{ij} \frac{1}{S} \sum_{s=1}^{S} \sum_k \theta_{jk}^s \phi_{kx_{ij}^{\text{test}}}^s \qquad \theta_{jk}^s = \frac{\alpha^s \pi_k^s + n_{jk\cdot}^s}{\alpha^s + n_{j\cdot\cdot}^s} \qquad \phi_{kw}^s = \frac{\beta\tau_w + n_{\cdot kw}^s}{\beta + n_{\cdot k\cdot}^s} \qquad (17)$$

We used all samples obtained by the Gibbs sampling algorithms after an initial burn-in period; each point in the predictive probabilities plots below is obtained from the samples collected thus far.

The results, shown in Figure 2, display a significant improvement in accuracy of CV-HDP over CV-LDA, both in terms of the bound on the training log-likelihood as well as for the test-set log-likelihood. This is caused by the fact that CV-HDP is learning the variational distributions over the hyperparameters. We note that we have not trained $\beta$ or $\tau$ for any of these methods. In fact, initial results for CV-HDP show no additional improvement in test-set log-likelihood, in some cases even a deterioration of the results. A second observation is that convergence of all variational methods is faster than for the sampling methods. Thirdly, we see significant local optima effects in our simulations. For example, G-HDP$_{100}$ achieves the best results, better than G-HDP$_1$, indicating that pruning topics is a better way than adding topics to escape local optima in these models and leads to better posterior modes.

In further experiments we have also found that the variational methods benefit from better initializations due to local optima. In Figure 3 we show results when the variational methods were initialized at the last state obtained by G-HDP$_{100}$. We see that indeed the variational methods were able to find significantly better local optima in the vicinity of the one found by G-HDP$_{100}$, and that CV-HDP is still consistently better than the other variational methods.

## 5   Discussion

In this paper we have explored collapsed variational inference for the HDP. Our algorithm is the first to deal with the HDP and with posteriors over the parameters of Dirichlet distributions. We found that the CV-HDP performs significantly better than the CV-LDA on both test-set likelihood and the variational bound. A caveat is that CV-HDP gives slightly worse test-set likelihood than collapsed Gibbs sampling. However, as discussed in the introduction, we believe there are advantages to variational approximations that are not available to sampling methods. A second caveat is that our variational approximation works only for two layer HDPs—a layer of group-specific DPs, and a global DP tying the groups together. It would be interesting to explore variational approximations for more general HDPs.

CV-HDP presents an improvement over CV-LDA in two ways. Firstly, we use a more sophisticated variational approximation that can infer posterior distributions over the higher level variables in the model. Secondly, we use a more sophisticated HDP based model with an infinite number of topics, and allow the model to find an appropriate number of topics automatically. These two advances are coupled, because we needed the more sophisticated variational approximation to deal with the HDP.

Along the way we have also proposed two useful technical tricks. Firstly, we have a new truncation technique that guarantees nesting. As a result we know that the variational bound on the marginal log-likelihood will reach its highest value (ignoring local optima issues) when $K \to \infty$. This fact should facilitate the search over number of topics or clusters, e.g. by splitting and merging topics, an aspect that we have not yet fully explored, and for which we expect to gain significantly from in the face of the observed local optima issues in the experiments. Secondly, we have an improved second-order approximation that is able to handle the often encountered digamma function accurately.

An issue raised by the reviewers and in need of more thought by the community is the need for better evaluation criteria. The standard evaluation criteria in this area of research are the variational bound

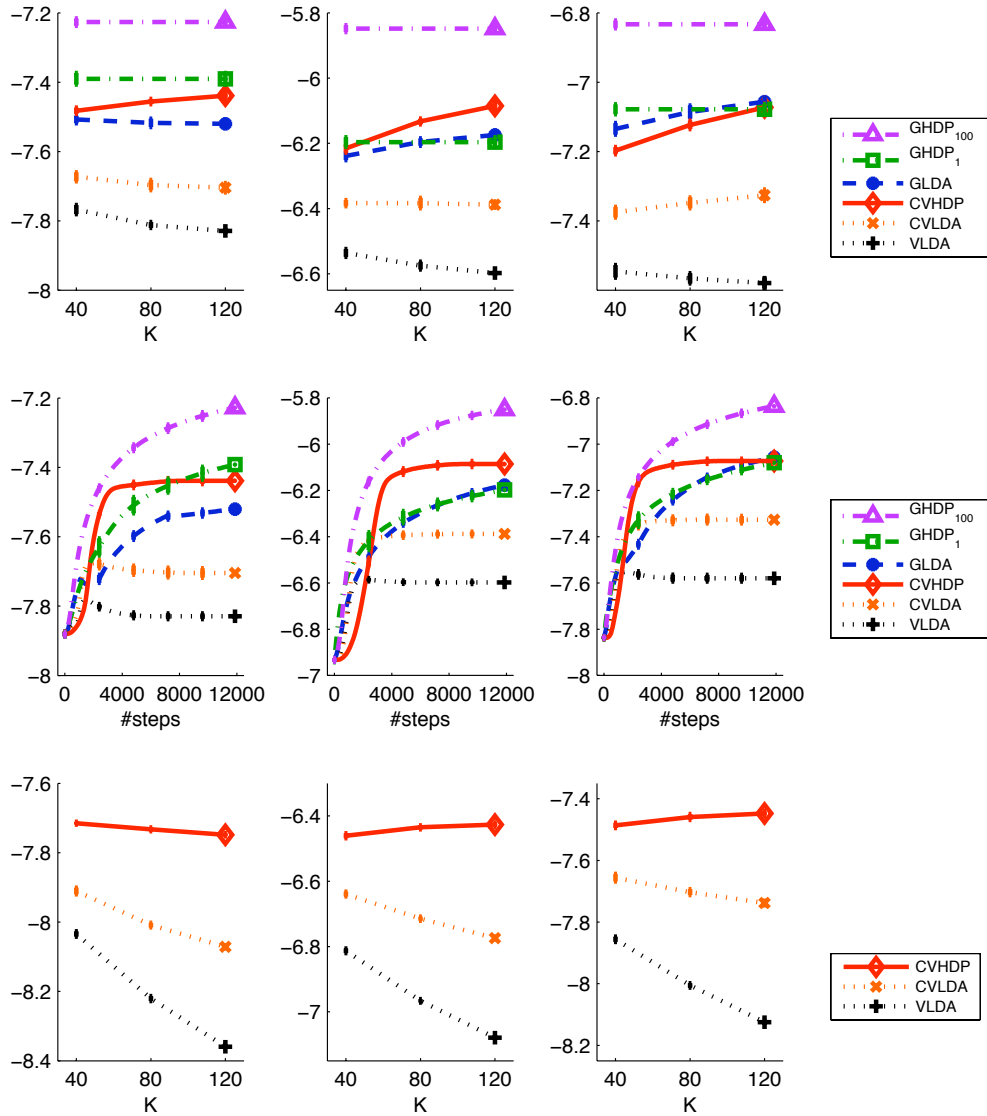

Figure 2: Left column: KOS, Middle column: Reuters and Right column: 20Newsgroups. Top row: $\log p(\mathbf{x}^{\text{test}})$ as a function of $K$, Middle row: $\log p(\mathbf{x}^{\text{test}})$ as a function of number of steps (defined as number of iterations multiplied by $K$) and Bottom row: variational bounds as a function of $K$. Log probabilities are on a per word basis. Shown are averages and standard errors obtained by repeating the experiments 10 times with random restarts. The distribution over the number of topics found by G-HDP$_1$ are: KOS: $K = 113.2 \pm 11.4$, Reuters: $K = 60.4 \pm 6.4$, 20News: $K = 83.5 \pm 5.0$. For G-HDP$_{100}$ we have: KOS: $K = 168.3 \pm 3.9$, Reuters: $K = 122.2 \pm 5.0$, 20News: $K = 128.1 \pm 6.6$.

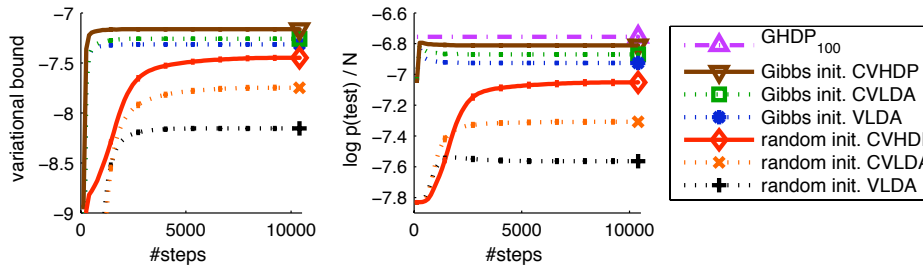

Figure 3: G-HDP$_{100}$ initialized variational methods ($K = 130$), compared against variational methods initialized in the usual manner with $K = 130$ as well. Results were averaged over 10 repeats.

and the test-set likelihood. However both confound improvements to the model and improvements to the inference method. An alternative is to compare the computed posteriors over latent variables on toy problems with known true values. However such toy problems are much smaller than real world problems, and inferential quality on such problems may be of limited interest to practitioners.

We expect the proliferation of Dirichlet-multinomial models and their many exciting applications to continue. For some applications variational approximations may prove to be the most convenient tool for inference. We believe that the methods presented here are applicable to many models of this general class and we hope to provide general purpose software to support inference in these models in the future.

## A   Variational lower bound

$$\mathbb{E}[\log p(\mathbf{z},\mathbf{x}|\alpha,\pi,\tau) - \log q(\mathbf{z})] - \mathrm{KL}[q(\alpha)\|p(\alpha)] - \mathrm{KL}[q(\beta)\|p(\beta)] - \sum_{k=1}^{K} \mathrm{KL}[q(\tilde{\pi}_k)\|p(\tilde{\pi}_k)] - \mathrm{KL}[q(\tau)\|p(\tau)] \tag{18}$$

$$= \sum_d \log \frac{\Gamma(\mathbb{E}[\alpha])}{\Gamma(\mathbb{E}[\alpha]+n_{d..})} + \sum_{dk} \mathbb{F}\left[\log \frac{\Gamma(\mathbb{G}[\alpha]\mathbb{G}[\pi_k]+n_{dk.})}{\Gamma(\mathbb{G}[\alpha]\mathbb{G}[\pi_k])}\right] + \sum_k \mathbb{F}\left[\log \frac{\Gamma(\mathbb{E}[\beta])}{\Gamma(\mathbb{E}[\beta]+n_{.k.})}\right] + \sum_{kw} \mathbb{F}\left[\log \frac{\Gamma(\mathbb{G}[\beta]\mathbb{G}[\tau_w]+n_{.kw})}{\Gamma(\mathbb{G}[\beta]\mathbb{G}[\tau_w])}\right]$$

$$- \log \frac{(b_\alpha - \sum_d \mathbb{E}[\log \eta_d])^{a_\alpha + \mathbb{E}[s..]}}{b_\alpha^{a_\alpha}} \frac{\Gamma(a_\alpha)}{\Gamma(a_\alpha + \mathbb{E}[s..])} \mathbb{G}[\alpha]^{\mathbb{E}[s..]} e^{\mathbb{E}[\alpha] \sum_d \mathbb{E}[\log \eta_d]} - \sum_{dk} \sum_{i=1}^{n_d} q(z_{id}=k) \log q(z_{id}=k)$$

$$- \log \frac{(b_\beta - \sum_k \mathbb{E}[\log \xi_k])^{a_\beta + \mathbb{E}[t..]}}{b_\beta^{a_\beta}} \frac{\Gamma(a_\beta)}{\Gamma(a_\beta + \mathbb{E}[t..])} \mathbb{G}[\beta]^{\mathbb{E}[t..]} e^{\mathbb{E}[\beta] \sum_k \mathbb{E}[\log \xi_k]}$$

$$- \sum_k \log \frac{\Gamma(1+\gamma+\mathbb{E}[s.k]+\mathbb{E}[s.>k])}{\gamma \Gamma(1+\mathbb{E}[s.k])\Gamma(\gamma+\mathbb{E}[s.>k])} \mathbb{G}[\tilde{\pi}_k]^{\mathbb{E}[s.k]} \mathbb{G}[1-\tilde{\pi}_k]^{\mathbb{E}[s.>k]} - \log \frac{\Gamma(\kappa+\mathbb{E}[t..])}{\Gamma(\kappa)} \prod_w \frac{\Gamma(\kappa\tau_w)}{\Gamma(\kappa\tau_w+\mathbb{E}[t.w])} \mathbb{G}[\tau_w]^{\mathbb{E}[t.w]}$$

where $\mathbb{F}[f(n)] = \mathbb{P}_+[n](f(\mathbb{E}_+[n]) + \frac{1}{2}\mathbb{V}_+[n]f''(\mathbb{E}_+[n]))$ is the improved second order approximation.

## Acknowledgements

We thank the reviewers for thoughtful and constructive comments. MW was supported by NSF grants IIS-0535278 and IIS-0447903.

## Footnotes

[1]In this paper, by HDP we shall mean the two level HDP topic model in Section 2. We do not claim to have derived a VB inference for the general HDP in [4], which is significantly more difficult; see final discussions.

[2]We actually set these values using a fixed but somewhat elaborate scheme which is the reason they ended up different for each dataset. Note that this scheme simply converts prior expectations about the number of topics and amount of sharing into hyperparameter values, and that they were never tweaked. Since they always ended up in these compact ranges and since we do not expect a strong dependence on their values inside these ranges we choose to omit the details.

## References

[1] R. G. Cowell, A. P. Dawid, S. L. Lauritzen, and D. J. Spiegelhalter. *Probabilistic Networks and Expert Systems*. Springer-Verlag, 1999.

[2] M. J. Beal and Z. Ghahramani. Variational Bayesian learning of directed graphical models with hidden variables. *Bayesian Analysis*, 1(4), 2006.

[3] D. M. Blei, A. Y. Ng, and M. I. Jordan. Latent Dirichlet allocation. *Journal of Machine Learning Research*, 3:993–1022, 2003.

[4] Y. W. Teh, M. I. Jordan, M. J. Beal, and D. M. Blei. Hierarchical Dirichlet processes. *Journal of the American Statistical Association*, 101(476):1566–1581, 2006.

[5] T. P. Minka and J. Lafferty. Expectation propagation for the generative aspect model. In *Proceedings of the Conference on Uncertainty in Artificial Intelligence*, volume 18, 2002.

[6] W. Buntine and A. Jakulin. Applying discrete PCA in data analysis. In *Proceedings of the Conference on Uncertainty in Artificial Intelligence*, volume 20, 2004.

[7] Y. W. Teh, D. Newman, and M. Welling. A collapsed variational Bayesian inference algorithm for latent Dirichlet allocation. In *Advances in Neural Information Processing Systems*, volume 19, 2007.

[8] D. M. Blei and M. I. Jordan. Variational inference for Dirichlet process mixtures. *Bayesian Analysis*, 1(1):121–144, 2006.

[9] K. Kurihara, M. Welling, and N. Vlassis. Accelerated variational DP mixture models. In *Advances in Neural Information Processing Systems*, volume 19, 2007.

[10] P. Liang, S. Petrov, M. I. Jordan, and D. Klein. The infinite PCFG using hierarchical Dirichlet processes. In *Proceedings of the Conference on Empirical Methods in Natural Language Processing*, 2007.

[11] J. Sethuraman. A constructive definition of Dirichlet priors. *Statistica Sinica*, 4:639–650, 1994.

[12] T.L. Griffiths and M. Steyvers. A probabilistic approach to semantic representation. In *Proceedings of the 24th Annual Conference of the Cognitive Science Society*, 2002.

[13] C. E. Antoniak. Mixtures of Dirichlet processes with applications to Bayesian nonparametric problems. *Annals of Statistics*, 2(6):1152–1174, 1974.

[14] M. J. Beal. *Variational Algorithms for Approximate Bayesian Inference*. PhD thesis, Gatsby Computational Neuroscience Unit, University College London, 2003.
